# Rate- and Phase-coded Autoassociative Memory

**Máté Lengyel**     **Peter Dayan**
Gatsby Computational Neuroscience Unit, University College London
17 Queen Square, London WC1N 3AR, United Kingdom
{lmate,dayan}@gatsby.ucl.ac.uk

## Abstract

Areas of the brain involved in various forms of memory exhibit patterns of neural activity quite unlike those in canonical computational models. We show how to use well-founded Bayesian probabilistic autoassociative recall to derive biologically reasonable neuronal dynamics in recurrently coupled models, together with appropriate values for parameters such as the membrane time constant and inhibition. We explicitly treat two cases. One arises from a standard Hebbian learning rule, and involves activity patterns that are coded by graded firing rates. The other arises from a spike timing dependent learning rule, and involves patterns coded by the phase of spike times relative to a coherent local field potential oscillation. Our model offers a new and more complete understanding of how neural dynamics may support autoassociation.

## 1   Introduction

Autoassociative memory in recurrently coupled networks seems fondly regarded as having been long since solved, at least from a computational perspective. Its neurobiological importance, as a model of episodic (event) memory storage and retrieval (from noisy and partial inputs) in structures such as area CA3 in the hippocampus, is of course clear [1]. This perhaps suggests that it is only the exact mapping of the models to the neural substrate that holds any remaining theoretical interest.

However, the characteristic patterns of activity in areas such as CA3 that are involved in memory are quite unlike those specified in the bulk of models. In particular neurons (for instance hippocampal place cells) show *graded* activity during recall [2], prominent theta frequency *oscillations* [3] and an apparent variety of rules governing synaptic plasticity [4, 5]. The wealth of studies of memory capacity of attractor networks of binary units does not give many clues to the specification, analysis or optimization of networks acting in these biologically relevant regimes. In fact, even theoretical approaches to autoassociative memories with graded activities are computationally brittle.

Here, we generalize previous analyses [6, 7] to address these issues. Formally, these models interpret recall as Bayesian inference based on information given by the noisy input, the synaptic weight matrix, and prior knowledge about the distribution of possible activity patterns coding for memories. More concretely (see section 2), the assumed activity patterns and synaptic plasticity rules determine the term in neuronal update dynamics that describes interactions between interconnected cells. Different aspects of biologically reasonable autoassociative memories arise from different assumptions. We show (section 3)

We thank Boris Gutkin for helpful discussions on the phase resetting characteristics of different neuron types. This work was supported by the Gatsby Charitable Foundation.

that for neurons are characterized by their graded firing rates, the regular rate-based charac-
terization of neurons effectively approximates optimal Bayesian inference. Optimal values
for parameters of the update dynamics, such as level of inhibition or leakage conductance,
are inherently provided by our formalism. We then extend the model (section 4) to a set-
ting involving spiking neurons in the context of a coherent local field potential oscillation
(LFPO). Memories are coded by the the phase of the LFPO at which each neuron fires, and
are stored by spike timing dependent plasticity. In this case, the biophysically plausible
neuronal interaction function takes the form of a phase reset curve: presynaptic firing ac-
celerates or decelerates the postsynaptic cell, depending on the relative timing of the two
spikes, to a degree that is proportional to the synaptic weight between the two cells.

## 2   MAP autoassociative recall

The first requirement is to specify the task for autoassociative recall in a probabilistically
sound manner. This specification leads to a natural account of the dynamics of the neurons
during recall, whose form is largely determined by the learning rule. Unfortunately, the
full dynamics includes terms that are not purely local to the information a post-synaptic
neuron has about pre-synaptic activity, and we therefore consider approximations that re-
store essential characteristics necessary to satisfy the most basic biological constraints. We
validate the quality of the approximations in later sections.

**The construction of the objective function:** Consider an autoassociative network which
has stored information about $M$ memories $\mathbf{x}^1 \ldots \mathbf{x}^M$ in a synaptic weight matrix, $\mathbf{W}$ be-
tween a set of $N$ neurons. We specify these quantities rather generally at first to allow for
different ways of construing the memories later. The most complete probabilistic descrip-
tion of its task is to report the conditional distribution $\mathrm{P}\left[\mathbf{x}|\tilde{\mathbf{x}}, \mathbf{W}\right]$ over the activities $\mathbf{x}$ *given*
noisy inputs $\tilde{\mathbf{x}}$ and the weights. The uncertainty in this posterior distribution has two roots.
First, the activity pattern referred to by the input is unclear unless there is no input noise.
Second, biological synaptic plasticity rules are data-lossy 'compression algorithms', and
so $\mathbf{W}$ specifies only imprecise information about the stored memories.

In an ideal case, $\mathrm{P}\left[\mathbf{x}|\tilde{\mathbf{x}}, \mathbf{W}\right]$ would have support only on the $M$ stored patterns $\mathbf{x}^1 \ldots \mathbf{x}^M$.
However, biological storage methods lead to weights $\mathbf{W}$ that permit a much greater range
of possibilities. We therefore consider methods that work in the full space of activities $\mathbf{x}$.
In order to optimize the probability of extracting just the correct memory, decision theory
encourages us to maximize the posterior probability [8]:

$$\hat{\mathbf{x}} := \operatorname{argmax}_{\mathbf{x}} \mathrm{P}\left[\mathbf{x}|\tilde{\mathbf{x}}, \mathbf{W}\right] \ , \ \ \mathrm{P}\left[\mathbf{x}|\tilde{\mathbf{x}}, \mathbf{W}\right] \propto \mathrm{P}\left[\mathbf{x}\right] \mathrm{P}\left[\tilde{\mathbf{x}}|\mathbf{x}\right] \mathrm{P}\left[\mathbf{W}|\mathbf{x}\right] \tag{1}$$

The first term in Eq.1 imports prior knowledge of the statistical characteristics of the memo-
ries, and is assumed to factorize: $\mathrm{P}\left[\mathbf{x}\right] := \prod_i \mathrm{P}_x\left[x_i\right]$. The second term describes the noise
process corrupting the inputs. For unbiased noise it will be a term in $\mathbf{x}$ that is effectively
centered on $\tilde{\mathbf{x}}$. We assume that the noise corrupting each element of the patterns is indepen-
dent, and independent of the original pattern, so $\mathrm{P}\left[\tilde{\mathbf{x}}|\mathbf{x}\right] := \prod_i \mathrm{P}\left[\tilde{x}_i|\mathbf{x}\right] := \prod_i \mathrm{P}\left[\tilde{x}_i|x_i\right]$.

The third term assesses the likelihood that the weight matrix came from a training set of
size $M$ including pattern $\mathbf{x}$.[1]  Biological constraints encourage consideration of learning
updates for the synapse from neuron $j$ to neuron $i$ that are *local* to the pre-synaptic $(x_j^m)$
and post-synaptic $(x_i^m)$ activities of connected neurons when pattern $\mathbf{x}^m$ is stored:

$$\Delta w_{i,j}^m := \Omega\left(x_i^m, x_j^m\right) \tag{2}$$

We assume the contributions of individual training patterns are additive, $W_{i,j} :=$
$\sum_m \Delta w_{i,j}^m$, and that there are no autapses in the network, $W_{i,i} := 0$.

Storing a single random pattern drawn from the prior distribution will result in a synaptic weight change with a distribution determined by the prior and the learning rule, having $\mu_{\Delta w} = \langle \Omega(x_1, x_2) \rangle_{P_x[x_1] \cdot P_x[x_2]}$ mean, and $\sigma^2_{\Delta w} = \langle \Omega^2(x_1, x_2) \rangle_{P_x[x_1] \cdot P_x[x_2]} - \mu^2_{\Delta w}$ variance. Storing $M-1$ random patterns means adding $M-1$ iid. random variables and thus, for moderately large $M$, results in a synaptic weight with an approximately Gaussian distribution $P[W_{i,j}] \simeq \mathcal{G}(W_{i,j}; \mu_W, \sigma_W)$, with mean $\mu_W = (M-1)\mu_{\Delta w}$ and variance $\sigma^2_W = (M-1)\sigma^2_{\Delta w}$. Adding a further *particular* pattern $\mathbf{x}$ is equivalent to adding a random variable with a mean determined by the learning rule, and zero variance, thus:

$$P[W_{i,j}|x_i, x_j] \simeq \mathcal{G}(W_{i,j}; \mu_W + \Omega(x_i, x_j), \sigma_W) \tag{3}$$

We also make the approximation that elements of the synaptic weight matrix are independent, and thus write: $P[\mathbf{W}|\mathbf{x}] := \prod_{i,j \neq i} P[W_{i,j}|x_i, x_j]$.

Having restricted our horizons to maximum a posteriori (MAP) inference, we can consider as an objective function the log of the posterior distribution. In the light of our factorizability assumptions, this is

$$
\begin{aligned}
O(\mathbf{x}) &= \log P[\mathbf{x}] + \log P[\tilde{\mathbf{x}}|\mathbf{x}] + \log P[\mathbf{W}|\mathbf{x}] \\
&= \sum_i \log P[x_i] + \sum_i \log P[\tilde{x}_i|x_i] + \sum_{i,j \neq i} \log P[W_{i,j}|x_i, x_j]
\end{aligned} \tag{4}
$$

**Neuronal update dynamics:** Finding the global maximum of the objective function, as stated in equation 1, is computationally extravagant, and biologically questionable. We therefore specify neuronal dynamics arising from gradient ascent on the objective function:

$$\tau_x \dot{\mathbf{x}} \propto \nabla_{\mathbf{x}} O(\mathbf{x}) \ . \tag{5}$$

Combining equations 4 and 5 we get

$$\tau_x \frac{dx_i}{dt} = \frac{\partial}{\partial x_i} \log P[\mathbf{x}] + \frac{\partial}{\partial x_i} \log P[\tilde{\mathbf{x}}|\mathbf{x}] + \frac{\partial}{\partial x_i} \log P[\mathbf{W}|\mathbf{x}], \text{where} \tag{6}$$

$$\frac{\partial}{\partial x_i} \log P[\mathbf{W}|\mathbf{x}] = \sum_{j \neq i} \frac{\partial}{\partial x_i} \log P[W_{i,j}|x_i, x_j] + \frac{\partial}{\partial x_i} \log P[W_{j,i}|x_j, x_i] \ . \tag{7}$$

The first two terms in equation 6 only depend on the activity of the neuron itself and its input. For example, for a Gaussian prior $P_x[x_i] = \mathcal{G}(W_{i,j}; \mu_x, \sigma_x)$ and unbiased Gaussian noise on the input $P[\tilde{x}_i|x_i] = \mathcal{G}(\tilde{x}_i; x_i, \sigma_{\tilde{x}})$, these would be:

$$\frac{d}{dx_i} \log P[x_i] + \frac{d}{dx_i} \log P[\tilde{x}_i|x_i] = \frac{\mu_x - x_i}{\sigma_x^2} + \frac{\tilde{x}_i - x_i}{\sigma_{\tilde{x}}^2} = \frac{\mu_x}{\sigma_x^2} - \left(\frac{1}{\sigma_x^2} + \frac{1}{\sigma_{\tilde{x}}^2}\right) x_i + \frac{\tilde{x}_i}{\sigma_{\tilde{x}}^2} \tag{8}$$

The first term on the right-hand side of the last equality expresses a constant bias; the second involves self-decay; and the third describes the effect of the input.

The terms in equation 7 indicate how a neuron should take into account the activity of other neurons based on the synaptic weights. From equation 3, the terms are

$$\frac{\partial}{\partial x_i} \log P[W_{i,j}|x_i, x_j] = \frac{1}{\sigma_W^2} \left[(W_{i,j} - \mu_W) \frac{\partial}{\partial x_i} \Omega(x_i, x_j) - \Omega(x_i, x_j) \frac{\partial}{\partial x_i} \Omega(x_i, x_j)\right] \tag{9}$$

$$\frac{\partial}{\partial x_i} \log P[W_{j,i}|x_j, x_i] = \frac{1}{\sigma_W^2} \left[(W_{j,i} - \mu_W) \frac{\partial}{\partial x_i} \Omega(x_j, x_i) - \Omega(x_j, x_i) \frac{\partial}{\partial x_i} \Omega(x_j, x_i)\right] \tag{10}$$

Two aspects of the above formulæ are biologically troubling. The last terms in each express the effects of other cells, but without there being corresponding synaptic weights. We approximate these terms using their mean values over the prior distribution. In this case $\alpha_i^+ = \langle \Omega(x_i, x_j) \frac{\partial}{\partial x_i} \Omega(x_i, x_j) \rangle_{P_x[x_j]}$ and $\alpha_i^- = \langle \Omega(x_j, x_i) \frac{\partial}{\partial x_i} \Omega(x_j, x_i) \rangle_{P_x[x_j]}$ contribute terms that only depend on the activity of the updated cell, and so can be lumped with the prior- and input-dependent terms of Eq.8.

Further, equation 10 includes synaptic weights, $W_{j,i}$, that are *post*synaptic with respect to the updated neuron. This would require the neuron to change its activity depending on the weights of its postsynaptic synapses. One simple work-around is to approximate

a postsynaptic weight by the mean of its conditional distribution given the corresponding presynaptic weight: $W_{j,i} \simeq \langle P[W_{j,i}|W_{i,j}] \rangle$. In the simplest case of perfectly symmetric or anti-symmetric learning, with $\Omega(x_i, x_j) = \pm \Omega(x_j, x_i)$, we have $W_{j,i} = \pm W_{j,i}$ and $\alpha_i^+ = \alpha_i^- = \alpha_i$. In the anti-symmetric case $\mu_w = 0$.

Making these assumptions, the neuronal interaction function simplifies to

$$H(x_i, x_j) = (W_{i,j} - \mu_W)\frac{\partial}{\partial x_i}\Omega(x_i, x_j) \tag{11}$$

and $\frac{2}{\sigma_W^2}\left[\sum_{j \neq i} H(x_i, x_j) - (N-1)\alpha_i\right]$ is the weight-dependent term of equation 7. Equation 11 shows that there is a simple relationship between the synaptic plasticity rule, $\Omega(x_i, x_j)$, and the neuronal interaction function, $H(x_i, x_j)$, that is approximately optimal for reading out the information that is encoded in the synaptic weight matrix by that synaptic plasticity rule. It also shows that the magnitude of this interaction should be proportional to the synaptic weight connecting the two cells, $W_{i,j}$.

We specialize this analysis to two important cases with (a) graded, rate-based, or (b) spiking, oscillatory phase-based, activities. We derive appropriate dynamics from learning rules, and show that, despite the approximations, the networks have good recall performance.

## 3   Rate-based memories

The most natural assumption about pattern encoding is that the activity of each unit is interpreted directly as its firing rate. Note, however, that most approaches to autoassociative memory assume *binary* patterns [9], sitting ill with the lack of saturation in cortical or hippocampal neurons in the appropriate regime. Experiments [10] suggest that regulating activity levels in such networks is very tricky, requiring exquisitely carefully tuned neuronal dynamics. There has been work on graded activities in the special case of line or surface attractor networks [11, 12], but these also pose dynamical complexitiese. By contrast, graded activities are straightforward in our framework.

Consider Hebbian covariance learning: $\Omega_{\mathrm{cov}}(x_i, x_j) := A_{\mathrm{cov}}(x_i - \mu_x)(x_j - \mu_x)$, where $A_{\mathrm{cov}} > 0$ is a normalizing constant and $\mu_x$ is the mean of the prior distribution of the patterns to be stored. The learning rule is symmetric, and so, based on Eq.11, the optimal neuronal interaction function is $H_{\mathrm{cov}}(x_i, x_j) = A_{\mathrm{cov}}(W_{i,j} - \mu_W)(x_j - \mu_x)$. This leads to a term in the dynamics which is the conventional weighted sum of pre-synaptic firing rates. The other key term in the dynamics is $\alpha_i = -A_{\mathrm{cov}}^2 \sigma_x^2 (x_i - \mu_x)$, where $\sigma_x^2$ is the variance of the prior distribution, expressing self-decay to a baseline activity level determined by $\mu_x$. The prior- and input-dependent terms also contribute to self-decay as shown in Eq.8. Integration of the weighted sum of inputs plus decay to baseline constitute the widely used leaky integrator reduction of a single neuron [10].

Thus, canonical models of synaptic plasticity (the Hebbian covariance rule) and single neuron firing rate dynamics are exactly matched for autoassociative recall. Optimal values for *all parameters* of single neuron dynamics (except the membrane time constant determining the speed of gradient ascent) are directly implied. This is important, since it indicates how to solve the problem for graded autoassociative memories (as opposed to saturing ones [14, 15]), that neuronal dynamics have to be finely tuned. As examples, the leak conductance is given by the sum of the coefficients of all terms linear in $x_i$, the optimal bias current is the sum of all terms independent of $x_i$, and the level of inhibition can be determined from the negative terms in the interaction function, $-\mu_W$ and $-\mu_x$.

Since our derivation embodies a number of approximations, we performed numerical simulations. To gauge the performance of the Bayes-optimal network we compared it to networks of increasing complexity (Fig. 1A,B). A trivial lower bound of performance is

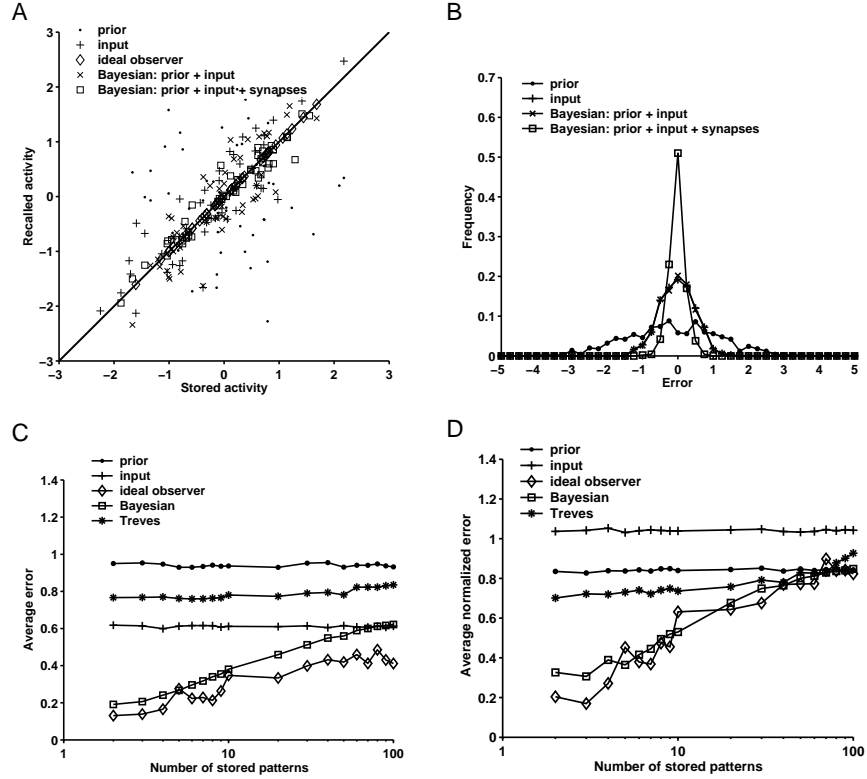

**Figure 1**: Performance of the rate-coded Bayesian inference network ($\square$), compared to a Bayesian network that only takes into account evidence from the prior and the input but not from the synaptic weight matrix ($\times$), a network that randomly generates patterns from the prior distribution ($\bullet$), a network that transmits its input to its output ($+$), and the 'ideal observer' having access to the list of stored patterns ($\Diamond$). **A.** Firing rates of single units at the end of the recall process (*y-axis*) against firing rates in the original pattern (*x-axis*). **B.** Frequency histograms of errors (difference between recalled and stored firing rates). The ideal observer is not plotted because its error distribution was a Dirac-delta at 0. **C, D.** Benchmarking the Bayesian network against the network of Treves [13] ($\ast$) on patterns of non-negative firing rates. Average error is the square root of the mean squared error (*C*), average normalized error measures only the angle difference between true and recalled activities (*D*). (These measures are not exactly the same as that used to derive the dynamics (equation 1), but are reasonably appropriate.) The prior distribution was Gaussian with $\mu_x = 0$ mean and $\sigma_x^2 = 1$ variance (*A,B*), or a Gaussian with $\mu_x = 0.5$ mean and $\sigma_x^2 = 0.25$ variance truncated below 0 (*C*) (yielding approximately $a = 0.5$ density), or ternary with $a = 0.5$ mean and density (*D*). The input was corrupted by unbiased Gaussian noise of $\sigma_{\tilde{x}}^2 = 1$ variance (*A,B*), or $\sigma_{\tilde{x}}^2 = 1.5$ variance (*C,D*) and cut at 0 (*C,D*). The learning rule was the covariance rule with $A_{\text{cov}} = 1$ (*A,B*), or with $A_{\text{cov}} = 1/Na^2$ (*C,D*). The number of cells in the network was $N = 50$ (*A,B*) and $N = 100$ (*C,D*), and the number of memories stored was $M = 2$ (*A,B*) or varied between $M = 2 \ldots 100$ (*C,D*, note logarithmic scale). For each data point, 10 different networks were simulated with a different set of stored patterns, and for each network, 10 attempts at recall were made, with a noisy version of a randomly chosen pattern as the input and with activities initialized at this input.

given by a network that generates random patterns from the same prior distribution from which the patterns to be stored were drawn ($P[\mathbf{x}]$). Another simple alternative is a network that simply transmits its input ($\tilde{\mathbf{x}}$) to its output. (Note that the 'input only' network is not necessarily superior to the 'prior only' network: their relative effectiveness depends on the relative variances of the prior and noise distributions, a narrow prior with a wide noise distribution would make the latter perform better, as in Fig. 1D). The Bayesian inference network performs considerably better than any of these simple networks. Crucially,

this improvement depends on the information encoded in synaptic weights: the network practically falls back to the level of the 'input only' network (or the 'prior only' network, whichever is the better, data not shown) if this information is ignored at the construction of the recall dynamics (by taking the third term in Eq. 6 to be 0).

An upper bound on the performance of any network using some biological form of synaptic plasticity comes from an 'ideal observer' which knows the complete list of stored patterns (rather than its distant reflection in the synaptic weight matrix) and computes and compares the probability that each was corrupted to form the input $\bar{x}$ to find the best match (rather than using neural dynamics). Such an ideal observer only makes errors when both the number of patterns stored and the noise in the input is sufficiently large, so that corrupting a stored pattern is likely to make it more similar to another stored pattern. In the case shown in Fig. 1A,B, this is not the case, since only two patterns were stored, and the ideal observer performs perfectly as expected. Nevertheless, there may be situations in which perfect performance is out of reach even for an ideal observer (Fig. 1C,D), which makes it a meaningful touchstone. In summary, the performance of any network can be assessed by measuring where it lies between the better one of the 'prior only' and 'input only' networks and the ideal observer.

As a further challenge, we also benchmarked our model against the model of Treves [13] (Fig. 1C,D), which we chose because it is a rare example of a network that was designed to have near optimal recall performance in the face of non-binary patterns. In this work, Treves considered *ternary* patterns, drawn from the distribution $\mathrm{P}\left[x_i\right] := \left(1 - \frac{4}{3}a\right)\delta\left(x_i\right) + a\delta\left(x_i - \frac{1}{2}\right) + \frac{a}{3}\delta\left(x_i - \frac{3}{2}\right)$, where $\delta\left(x\right)$ is the Dirac-delta function. Here, $a = \mu_x$ quantifies the *density* of the patterns (i.e. how non-sparse they are). The patterns are stored using the covariance rule as stated above (with $A_{\mathrm{cov}} := \frac{1}{Na^2}$). Neuronal update in the model is discrete, asynchronous, and involves two steps. First the 'local field' is calculated as $h_i := \sum_{j \neq i} W_{i,j} x_j - k\left(\sum_i x_i - N\right)^3 + \mathrm{Input}$, then the output of the neuron is calculated as a threshold linear function of the local field: $x_i := g\left(h_i - h_{\mathrm{Thr}}\right)$ if $h_i > h_{\mathrm{Thr}}$ and $x_i := 0$ otherwise, where $g := 0.53\, a/\left(1 - a\right)$ is the gain parameter, and $h_{\mathrm{Thr}} := 0$ is the threshold, and the value of $k$ is set by iterative search to optimize performance.

The comparison between Treves' network as we implemented it and our network is imperfect, since the former is optimized for recalling ternary patterns while, in the absence of neural evidence for ternary patterns, we used the simpler and more reasonable neural dynamics for our network that emerge from an assumption that the distribution over the stored patterns is Gaussian. Further, we corrupted the inputs by unbiased additive Gaussian noise (with variance $\sigma_{\bar{x}}^2 = 1.5$), but truncated the activities at 0, though did not adjust the dynamics of our network in the light of the truncation. Of course, these can only render our network *less* effective. Still, the Bayesian network clearly outperformed the Treves network when the patterns were drawn from a truncated Gaussian (Fig. 1C). The performance of the Bayesian network stayed close to that of an ideal observer assuming non-truncated Gaussian input, showing that most of the errors were caused by this assumption and not from suboptimality of neural interactions decoding the information in synaptic weights. Despite extensive efforts to find the optimal parameters for the Treves network, its performance did not even reach that of the 'input only' network.

Finally, again for ternary patterns, we also considered only penalizing errors about the *direction* of the vectors of recalled activities ignoring errors about their *magnitudes* (Fig. 1D). The Treves network did better in this case, but still not as well as the Bayesian network. Importantly, in both cases, in the regime where synaptic weights were saturated in the $M \to N$ limit and thus it was no longer possible to extract any useful information from the synaptic weights, the Bayesian network still only fell back to the level of the 'prior only' network, but the Treves network did not seem to have any such upper bound on its errors.

# 4 Phase-based memories

Brain areas known to be involved in memory processing demonstrate prominent oscillations (LFPOs) under a variety of conditions, including both wake and sleep states [16]. Under these conditions, the phases of the spikes of a neuron relative to the LFPO have been shown to be carefully controlled [17], and even to convey meaningful stimulus information, e.g. about the position of an animal in its environment [3] or retrieved odor identity [18]. The discovery of spike timing dependent plasticity (STDP) in which the *relative timing* of pre- and postsynaptic firings determines the sign and extent of synaptic weight change, offered new insights into how the information represented by spike times may be stored in neural networks [19]. However, bar some interesting suggestions about neuronal resonance [20], it is less clear how one might correctly recall information thereby stored in the synaptic weights.

The theory laid out in Section 2 allows us to treat this problem systematically. First, neuronal activities, $x_i$, will be interpreted as firing times relative to a reference phase of the ongoing LFPO, such as the peak of theta oscillation in the hippocampus, and will thus be circular variables drawn from a circular Gaussian. Next, our learning rule is an exponentially decaying Gabor-function of the phase difference between pre- and postsynaptic firing: $\Omega_{\mathrm{STDP}}(x_i, x_j) := A_{\mathrm{STDP}} \exp[\kappa_{\mathrm{STDP}} \cos(\Delta\phi_{i,j})] \sin(\Delta\phi_{i,j} - \phi_{\mathrm{STDP}})$ with $\Delta\phi_{i,j} = 2\pi(x_i - x_j)/T_{\mathrm{STDP}}$. STDP characteristics in different brain regions are well captured by this general formula, but the parameters determining their exact shapes greatly differ among regions. We constrain our analysis to the antisymmetric case, so that $\phi_{\mathrm{STDP}} = 0$, and set other parameters to match experimental data on hippocampal STDP [5]. The neuronal interaction function that satisfies Eq.11 is $\mathrm{H}_{\mathrm{STDP}}(x_i, x_j) = 2\pi A_{\mathrm{STDP}}/T_{\mathrm{STDP}} W_{i,j} \exp[\kappa_{\mathrm{STDP}} \cos(\Delta\phi_{i,j})] \left[\cos(\Delta\phi_{i,j}) - \kappa_{\mathrm{STDP}} \sin^2(\Delta\phi_{i,j})\right]$. This interaction function decreases firing phase, and thus accelerates the postsynaptic cell if the presynaptic spike precedes postsynaptic firing, and delays the postsynaptic cell if the presynaptic spike arrives just after the postsynaptic cell fired. This characteristic is the essence of the biphasic phase reset curve of type II cells [21], and has been observed in various types of neurons, including neocortical cells [22]. Thus again, our derivation directly couples STDP, a canonical model of synaptic plasticity, and phase reset curves in a canonical model of neural dynamics.

Numerical simulations tested again the various approximations. Performance of the network is shown in Fig.2 as is comparable to that of the rate coded network (Fig.2). Further simulations will be necessary to map out the performance of our network over a wider range of parameters, such as the signal-to-noise ratio.

# 5 Discussion

We have described a Bayesian approach to recall in autoassociative memories. This permits the derivation of neuronal dynamics appropriate to a synaptic plasticity rule, and we used this to show a coupling between canonical Hebbian and STDP plasticity rules and canonical rate-based and phase-based neuronal dynamics respectively. This provides an unexpectedly close link between optimal computations and actual implementations. Our method also leads to networks that are highly competent at recall.

There are a number of important direction for future work. First, even in phase-based networks, not all neurons fire in each period of the oscillation. This suggests that neurons may employ a dual code – the more rate-based probability of being active in a cycle, and the phase-based timing of the spike relative to the cycle [24]. The advantages of such a scheme have yet to be fully characterized.

Second, in the present framework the choice of the learning rule is arbitrary, as long as the

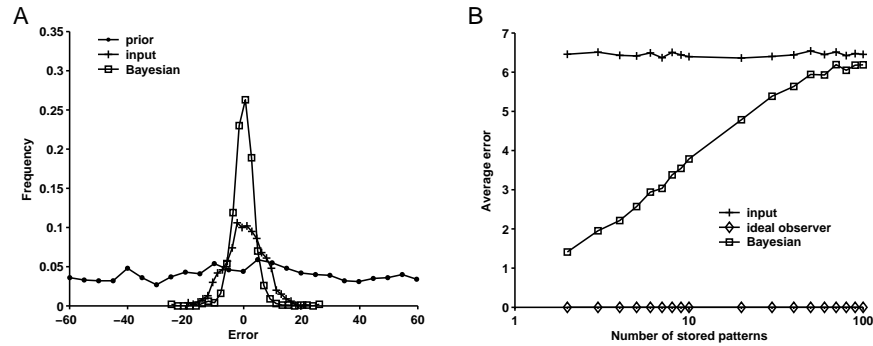

**Figure 2**: Performance of the phase-coded network. Error distribution for the ideal observer was a Dirac-delta at 0 (*B*) and was thus omitted from *A*. Average error of the 'prior only' network was too large (*A*) to be plotted in *B*. The prior was a von Mises distribution with $\mu_x = 0$ mean, $\kappa_x = 0.5$ concentration on a $T_\Theta = 125$ ms long cycle matching data on theta frequency modulation of pyramidal cell population activity in the hippocampus [23]. Input was corrupted by unbiased circular Gaussian (von Mises) noise with $\kappa_x = 10$ concentration. Learning rule was circular STDP rule with $A_{\text{STDP}} = 0.03$, $\kappa_{\text{STDP}} = 4$ and $T_{\text{STDP}} = T_\Theta$ parameters matching experimental data on hippocampal STDP [5] and theta periodicity. The network consisted of $N = 100$ cells, and the number of memories stored was $M = 10$ (*A*) or varied between $M = 2 \ldots 100$ (*B*, note logarithmic scale). For further explanation of symbols and axes, see Figure 1.

recall dynamics is optimally matched to it. Our formalism also suggests that there may be a way to optimally choose the learning rule itself in the first place, by matching it to the prior distribution of patterns. This approach would thus be fundamentally different from those seeking 'globally' optimal learning rules [25], and may be more similar to those used to find optimal tuning curves appropriately matching stimulus statistics [26].

## Footnotes

[1]Uncertainty about $M$ could also be incorporated into the model, but is neglected here.

## References

[1] Marr D. Philos Trans R Soc Lond B Biol Sci 262:23, 1971.
[2] O'Keefe J. Exp Neurol 51:78, 1976.
[3] O'Keefe J, Recce ML. Hippocampus 3:317, 1993.
[4] Bliss TVP, Lømo T. J Physiol (Lond) 232:331, 1973.
[5] Bi GQ, Poo MM. J Neurosci 18:10464, 1998.
[6] MacKay DJC. In Maximum entropy and Bayesian methods, 237, 1990.
[7] Sommer FT, Dayan P. IEEE Trans Neural Netw 9:705, 1998.
[8] Jaynes ET. Probability theory: the logic of science. Cambridge University Press, 2003.
[9] Amit DJ. Modeling brain function. Cambridge University Press, 1989.
[10] Dayan P, Abbott LF. Theoretical neuroscience. MIT Press, 2001.
[11] Zhang K. J Neurosci 16:2112, 1996.
[12] Seung HS. Proc Natl Acad Sci USA 93:13339, 1996.
[13] Treves A. Phys Rev A 42:2418, 1990.
[14] Hopfield JJ. Proc Natl Acad Sci USA 76:2554, 1982.
[15] Hopfield JJ. Proc Natl Acad Sci USA 81:3088, 1984.
[16] Buzsáki Gy. Neuron 33:325, 2002.
[17] Harris KD, et al. Nature 424:552, 2003.
[18] Li Z, Hopfield JJ. Biol Cybern 61:379, 1989.
[19] Abbott LF, Nelson SB. Nat Neurosci 3:1178, 2000.
[20] Scarpetta S, et al. Neural Comput 14:2371, 2002.
[21] Ermentrout B, et al. Neural Comput 13:1285, 2001.
[22] Reyes AD, Fetz FE. J Neurophysiol 69:1673, 1993.
[23] Klausberger T, et al. Nature 421:844, 2003.
[24] Mueller R, et al. In BioNet'96 , 70, 1976.
[25] Gardner E, Derrida B. J Phys A 21:271, 1988.
[26] Laughlin S. Z Naturforsch 36:901, 1981.
